# A Convergence Analysis of Log-Linear Training

**Simon Wiesler**
Computer Science Department
RWTH Aachen University
52056 Aachen, Germany
wiesler@cs.rwth-aachen.de

**Hermann Ney**
Computer Science Department
RWTH Aachen University
52056 Aachen, Germany
ney@cs.rwth-aachen.de

## Abstract

Log-linear models are widely used probability models for statistical pattern recognition. Typically, log-linear models are trained according to a convex criterion. In recent years, the interest in log-linear models has greatly increased. The optimization of log-linear model parameters is costly and therefore an important topic, in particular for large-scale applications. Different optimization algorithms have been evaluated empirically in many papers. In this work, we analyze the optimization problem analytically and show that the training of log-linear models can be highly ill-conditioned. We verify our findings on two handwriting tasks. By making use of our convergence analysis, we obtain good results on a large-scale continuous handwriting recognition task with a simple and generic approach.

## 1 Introduction

Log-linear models, also known as maximum entropy models or multiclass logistic regression, have found a wide range of applications in machine learning. Special cases of log-linear models include logistic regression for binary class problems and conditional random fields [10] for structured data, in particular sequential data. In recent years, the interest in log-linear models has increased greatly. Different models of log-linear form have been applied to natural language processing tasks, e.g. for segmentation [10], parsing [21], and information extraction [16], and many other tasks.

The most frequently mentioned advantages of log-linear models are, first, their discriminative nature, and second, the possibility to use arbitrary and correlated features in log-linear models. Furthermore, the conventional training of log-linear models is a strictly convex optimization problem. Thus, the global optimum of the training criterion is unique and no other local optima exist. Steepest descent and other gradient-based optimization algorithms are guaranteed to converge to the unique global optimum from any initialization. The probabilistic approach of log-linear models is beneficial in many practical applications. For example, log-linear models are directly defined as multiclass models and can be integrated into more complex classifiers.

For large datasets, the costs of training log-linear models are very high and limit their application range. Therefore, the efficient optimization of log-linear models is of great interest. The most widely used algorithms for this problem can be divided into three categories. *Bound optimization algorithms*, as generalized iterative scaling (GIS) [4] and variants of GIS have been used in earlier works. Later it has been found by several authors [17, 14, 21] that these algorithms converge very slowly and are inferior to gradient-based optimization algorithms. *First-order optimization algorithms* require the evaluation of the gradient of the objective function. The simplest algorithm of this category is steepest descent. The more sophisticated conjugate gradient (CG) and L-BFGS are now the standard choices for the training of log-linear models. *Newton's method* converges rapidly in a neighborhood of the optimum. For large-scale problems it is in general not applicable, because it requires the evaluation and storage of the Hessian matrix.

So far, a rigorous mathematical analysis of the optimization problem encountered in training of log-linear models has been missing. From optimization theory it is known that the convergence rate of first-order optimization algorithms depends on the condition number of the Hessian matrix at the optimum.[1] The dependence of the convergence behavior on the condition number is strongest for steepest descent. For high condition numbers, steepest descent is useless in practice [3, Chapter 9.3]. It can be shown that more sophisticated gradient-based optimization algorithms as CG and L-BFGS depend on the condition number as well [18, Chapter 5.1],[18, Chapter 9.1]. Apart from numerical reasons, the convergence behavior of Newton's method is completely independent of the condition number. In practice, it is not, because computing Newton's search direction requires solving a system of linear equations, which is more difficult for problems with high condition number [3, Chapter 9.5].

In this paper, we derive an estimate for the condition number of the objective function used for training of log-linear models. Our analysis shows that convergence can be accelerated by feature transformations. We verify our analytic results on two classification tasks. One is a small digit recognition task, the other a large-scale continuous handwriting recognition task with real-life data. The experiments show that in extreme cases, log-linear training can be so ill-conditioned that a usable model can only be found from a reasonable initialization. On the other hand, when care is taken, we obtain good results with a conceptually simple and generic approach.

The remaining paper is structured as follows: In the next section, we introduce the log-linear model and the training criterion. In Section 3, we give an overview on related work. Our novel convergence analysis is presented in Section 4. Experimental results are reported in Section 5. In the last section, we discuss our results.

## 2  Model Definition and Training Criterion

In this section, the log-linear model is defined and the necessary notation is introduced. Let $\mathcal{X} \subset \mathbb{R}^d$ denote the observation space and $\mathcal{C} = \{1, \ldots, C\}$ a finite set of classes. A *log-linear model* with parameters $\Lambda \in \mathbb{R}^{d \times C} = (\lambda_1; \ldots; \lambda_C)$ is a model for class-posterior probabilities of the form

$$p_\Lambda(c|x) = \frac{\exp(\lambda_c^T x)}{\sum_{c' \in \mathcal{C}} \exp(\lambda_{c'}^T x)} \ . \tag{1}$$

A log-linear model induces a decision rule via

$$r : \mathcal{X} \to \mathcal{C}, \quad x \mapsto \operatorname*{argmax}_{c \in \mathcal{C}} p_\Lambda(c|x) = \operatorname*{argmax}_{c \in \mathcal{C}} \lambda_c^T x. \tag{2}$$

The decision boundaries of log-linear models are linear. Non-linear decision boundaries can be achieved by embedding observations into a higher dimensional space. The *penalized maximum likelihood criterion* is regarded as the natural training criterion for log-linear models. Let $(x_n, c_n)_{n=1,\ldots,N}$ denote the training sample. Then the training criterion of log-linear models is an unconstrained optimization problem of the form

$$\hat{\Lambda} = \operatorname*{argmin}_{\Lambda \in \mathbb{R}^{d \times C}} \mathcal{F}(\Lambda), \text{with } \mathcal{F} : \mathbb{R}^{d \times C} \to \mathbb{R}, \quad \Lambda \mapsto -\frac{1}{N} \sum_{n=1}^{N} \log p_\Lambda(c_n|x_n) + \frac{\alpha}{2} \|\Lambda\|_2^2 \tag{3}$$

Here, $\mathcal{F}$ is the *objective function*, and $\alpha \geq 0$ the regularization constant. In the following, we refer to the optimization of the parameters of log-linear models as *log-linear training*.

The first and second partial derivatives of the objective function for $1 \leq c, \bar{c} \leq C$ and $1 \leq j, \bar{j} \leq d$ are:

$$\frac{\partial \mathcal{F}}{\partial \lambda_{c,j}}(\Lambda) \quad = \frac{1}{N} \sum_{n=1}^{N} \left( p_\Lambda(c|x_n) - \delta(c, c_n) \right) x_{n,j} + \alpha \lambda_{c,j} \ , \tag{4}$$

$$\frac{\partial^2 \mathcal{F}}{\partial \lambda_{c,j} \partial \lambda_{\bar{c},\bar{j}}}(\Lambda) \quad = \frac{1}{N} \sum_{n=1}^{N} p_\Lambda(c|x_n)(\delta(c, \bar{c}) - p_\Lambda(\bar{c}|x_n)) \, x_{n,j} x_{n,\bar{j}} + \alpha \, \delta(c, \bar{c})\delta(j, \bar{j}) \ . \tag{5}$$

Here, $\delta$ denotes the Kronecker delta. It can be shown that the Hessian matrix of $\mathcal{F}$ is positive semidefinite, and strictly positive definite for $\alpha > 0$. Thus, the optimization problem (3) is convex, respectively strictly convex (see e.g. [22]).

# 3   Related Work

In earlier works, e.g. [16, 10], the optimization problem (3) has been solved with generalized iterative scaling (GIS) [4] or improved iterative scaling [10]. Since then, it has been shown in several works that gradient-based optimization algorithms are far superior to iterative scaling methods. Minka [17] showed for logistic regression that iterative scaling methods perform poorly in comparison to conjugate gradient (CG). Although Minka performed his experiments only on artificial data with quite low dimensional features and a small number of observations, other authors came to similar findings. Malouf [14] performed experiments with (multiclass) log-linear models on typical natural language processing tasks. As Minka, he found that CG outperforms iterative scaling methods. Furthermore, he obtained best results with L-BFGS [12], which today is considered as the best algorithm for log-linear training. One of the first applications of CRFs to large-scale problems is by Sha and Pereira [21]. They confirmed again that L-BFGS is superior to CG and far superior to GIS.

All of the above mentioned papers concentrated on the empirical comparison of the performance of various optimization algorithms. The theoretical analysis of the optimization problem is very limited. Salakhutdinov [20] derived a convergence analysis for bound optimization algorithms as GIS and showed that GIS converges extremely slowly when features are highly correlated and are far from the origin. The disadvantage of Salakhutdinov's analysis is that, for log-linear models, it concerns only GIS which now is known to perform very badly in practice. The effect of correlation on the difficulty of the optimization problem has been noted by several authors, though not analyzed in detail, e.g. by Minka [17].

An interesting connection is the convergence analysis by LeCun et al. for neural network training [11]. Their analysis differs in a number of aspects from our analysis. Interestingly, we come to similar conclusions for the convergence behavior of log-linear training as LeCun et al. for neural network training. A comparison to their work is given in the discussion.

# 4   Convergence Analysis of Log-Linear Model Training

This section contains our theoretical result. We derive an estimate of the eigenvalues of the Hessian of log-linear training, which determine the convergence behavior of gradient-based optimization algorithms. First, we calculate the eigenvalues of the Hessian in terms of the eigenvalues of the uncentered covariance matrix. Our new Theorems 1 and 2 give lower and upper bounds for the condition number of the uncentered covariance matrix. The analysis of the case with regularization is based on the analysis of the unregularized case.

## 4.1   The Unregularized Case

Let $\Lambda^*$ be the limit of the optimization algorithm applied to problem (3) without regularization ($\alpha = 0$). The Hessian matrix of the objective function at the optimum depends on the posterior probabilities $p_{\Lambda^*}(c|x)$, which are of course unknown. In the following, we consider a simpler problem. We derive the eigenvalues of the Hessian at $\Lambda_0 = 0$. If the quadratic approximation of $\mathcal{F}$ at $\Lambda_0$ is good, the Hessian does not change strongly from $\Lambda_0$ to $\Lambda^*$, and the eigenvalues of $H_\mathcal{F}(\Lambda_0)$ are close to those of $H_\mathcal{F}(\Lambda^*)$. This enables us to draw conclusions about the convergence behavior of gradient-based optimization algorithms. The experiments in Section 5 justify our assumption. All experimental results are in accordance to the theoretical results.

For $\Lambda_0 = 0$, the posterior probabilities are uniform, i.e. $p_{\Lambda_0}(c|x) = C^{-1}$. Hence,

$$\frac{\partial^2 \mathcal{F}}{\partial \lambda_{c,j} \partial \lambda_{\bar{c},\bar{j}}}(\Lambda_0) = C^{-1} \left( \delta(c,\bar{c}) - C^{-1} \right) \; \frac{1}{N} \sum_{n=1}^{N} x_{n,j} x_{n,\bar{j}} \,. \tag{6}$$

The Hessian matrix can be written as a Kronecker product (see e.g. [8]): $H_\mathcal{F}(\Lambda_0) = S \otimes X$ . Here, $S \in \mathbb{R}^{C \times C}$ is defined by $S = C^{-1} \left( I_C - C^{-1} \mathbf{1}_C \right)$ , where $I_C \in \mathbb{R}^{C \times C}$ is the identity matrix, and $\mathbf{1}_C \in \mathbb{R}^{C \times C}$ denotes the matrix, where all entries are equal to one. The matrix $X \in \mathbb{R}^{d \times d}$ is the

uncentered covariance matrix: $X = \frac{1}{N} \sum_{n=1}^{N} x_n x_n^T$ . The eigenvalues of $S$ can be computed easily:

$$\mu_1(S) = 0, \quad \mu_2(S) = C^{-1} . \tag{7}$$

Let $0 \leq \mu_1(X) \leq \ldots \leq \mu_d(X)$ denote the eigenvalues of $X$. The eigenvalues of the Kronecker product $S \otimes X$ are of the form $\mu_i(S)\mu_j(X)$ (see [8, Theorem 4.2.12]). Therefore, the spectrum of the Hessian is determined by the eigenvalues of $X$:

$$\sigma(H_{\mathcal{F}}(\Lambda_0)) = \{0\} \cup \{C^{-1}\mu_1(X), \ldots, C^{-1}\mu_d(X)\} . \tag{8}$$

A difficulty in the analysis of the unregularized case is that the objective function is only convex, but not strictly convex. This is caused by the invariances of log-linear models. For instance, shifting all parameter vectors by a constant does not change the posterior probabilities. In addition, singularities appear as a result of linear dependencies in the features. Thus, one of the eigenvalues of the Hessian at the optimum is zero and the condition number is not defined. Intuitively, the convergence rate should not depend on the eigenvalue zero, since the objective function is constant in the direction of the corresponding eigenvectors. The classic proof about the convergence rate of steepest descent for quadratic functions with the Kantorovich inequality (see [13, p218]) can directly be generalized to the singular case. The convergence rate depends on the ratio of the largest and the smallest non-zero eigenvalue. Because of space constraints we omit this proof here. An analog result was shown by Notay [19] for the application of CG for solving systems of linear equations, which is equivalent to the minimization of quadratic functions. All results about the convergence behavior of conjugate gradient extend to the singular case, if instead of the complete spectrum only the non-zero eigenvalues are considered. Therefore, Notay defines the condition number of a singular matrix as the ratio of its largest eigenvalue and its smallest non-zero eigenvalue. In the following, we adopt this definition of the condition number. The condition number of the Hessian is then:

$$\kappa(H_{\mathcal{F}}(\Lambda_0)) = \kappa(X) = \frac{\mu_d(X)}{\min_{i:\mu_i(X)\neq 0} \mu_i(X)} . \tag{9}$$

In the following subsection, we analyze the condition number $\kappa(X)$.

## 4.2 The Eigenvalues of $X$

The dependence of the convergence behavior on the properties of $X$ is in accordance to experimental observations. Other researchers have noted before, that the use of correlated features leads to slower convergence [21]. Minka [17] noted that convergence slows down when adding a constant to the features, because this "introduces correlation, in the sense that" $X$ "has significant off-diagonals.". How can we verify these findings formally? The following theorem concerns the case of uncorrelated features. The proof is an application of Weyl's inequalities (see [9, Theorem 4.3.7]).

**Theorem 1.** *Suppose the features $x_i, 1 \leq i \leq d$, are uncorrelated with respect to the empirical distribution. Let $\mu_i$ and $\sigma_i^2$ denote the empirical mean and variance of $x_i$ for $1 \leq i \leq d$. Without loss of generality, we assume that the features are ordered such that $\sigma_1^2 \leq \ldots \leq \sigma_d^2$. Then the condition number of $X = \frac{1}{N} \sum_{n=1}^{N} x_n x_n^T$ is bounded by*

$$\frac{\max\{\sigma_1^2 + \|\mu\|_2^2, \sigma_d^2 + \mu_d^2\}}{\min\{\sigma_2^2, \sigma_1^2 + \mu_1^2\}} \leq \kappa(X) \leq \frac{\sigma_d^2 + \|\mu\|_2^2}{\sigma_1^2} . \tag{10}$$

*Proof of Theorem 1.* Since the features are uncorrelated, we have

$$X = \mathrm{diag}(\sigma_1^2, \ldots, \sigma_d^2) + \mu\mu^T \stackrel{\mathrm{def}}{=} A + B . \tag{11}$$

The eigenvalues of the sum of two Hermitian matrices can be estimated with Weyl's inequalities. Let $\lambda_j(M)$ denote the $j$-th eigenvalue in ascending order of a Hermitian $d \times d$-matrix $M$. Weyl's inequalities state that for all Hermitian $d \times d$-matrices $A, B$ and all $j, k$:

$$\lambda_{j+k-d}(A + B) \leq \lambda_j(A) + \lambda_k(B) , \tag{12}$$
$$\lambda_{j+k-1}(A + B) \geq \lambda_j(A) + \lambda_k(B) . \tag{13}$$

The eigenvalues of $A$ are the diagonal elements $\lambda_j(A) = \sigma_j^2$. $B$ is a rank-one matrix with the eigenvalues $\lambda_d(B) = \|\mu\|_2^2$ and $\lambda_j(B) = 0$ for $1 \leq j \leq d - 1$. The bounds for $\kappa(X)$ follow

with the application of (13) and (12) to the smallest and largest eigenvalue. For instance, the upper bound on the condition number follows from the application of (12) with $j = k = d$ to the largest eigenvalue and (13) with $j = k = 1$ to the lowest eigenvalue. The proof of the lower bound is analogous. The bound is sharpened by using the fact that every diagonal element of $X$ is an upper bound for the smallest eigenvalue and a lower bound for the largest eigenvalue (see [9, p181]). □

Analyzing the general case of correlated and unnormalized features is more difficult. The idea of the following theorem is regarding the off-diagonals as a perturbation of the diagonal matrix. This case can be analyzed with Geršgorin's circle theorem [9, Theorem 6.1.1], which states that all eigenvalues lie in circles around the diagonal entries of the matrix.

**Theorem 2.** *Let $\mu_i$ and $\sigma_i^2$ denote the empirical mean and variance of $x_i$ for $1 \leq i \leq d$ and assume that $\sigma_1^2 \leq \ldots \leq \sigma_d^2$. Let*

$$R_i = \sum_{j, j \neq i} |\text{Cov}(x_j, x_i)| \qquad (14)$$

*denote the radius of the $i$-th Geršgorin circle. Then, the largest and smallest eigenvalues of $X = \frac{1}{N} \sum_{n=1}^{N} x_n x_n^T$ are bounded by:*

$$\sigma_1^2 - R_1 \quad \leq \lambda_1(X) \quad \leq \min\{\sigma_1^2 + \mu_1^2, \sigma_d^2 + R_d\}, \qquad (15)$$
$$\max\{\sigma_d^2 + \mu_d^2, \sigma_1^2 - R_1 + \|\mu\|_2^2\} \quad \leq \lambda_d(X) \quad \leq \sigma_d^2 + R_d + \|\mu\|_2^2. \qquad (16)$$

The proof of Theorem 2 is a direct generalization of Theorem 1. In contrast to Theorem 1, only the bounds for the eigenvalues of $A$ obtained by Geršgorin's theorem are known instead of the exact eigenvalues. For strongly correlated features, the eigenvalues can be distributed almost arbitrarily according to the bounds (15) and (16). For weakly correlated features, the bounds are tighter. In particular, for normalized features and $R_1 < 1$, Theorem 2 implies:

$$1 \leq \kappa(X) \leq \frac{1 + R_d}{1 - R_1}. \qquad (17)$$

This shows that the best conditioning of the optimization problem is obtained for uncorrelated and normalized features. Conversely, our analysis shows that log-linear training can be accelerated by decorrelating the features and normalizing their means and variances, i.e. after *whitening* of the data.

### 4.3 The Regularized Case

In the following, we investigate the regularized training criterion, i.e. the objective function (3) with $\alpha > 0$. Since the Hessian of the $\ell_2$-regularization term is a multiple of the identity, the eigenvalues of the regularization term and the loss-term can be added. This has an important consequence. In the unregularized case, all non-zero eigenvalues depend on the eigenvalues of $X$. In the regularized case, the eigenvalue zero changes to $\alpha$, which is then the smallest non-zero eigenvalue of the Hessian. Therefore, the condition number depends only on the largest eigenvalue of $X$

$$\kappa(H_\mathcal{F}(\Lambda_0)) = \frac{C^{-1} \mu_d(X) + \alpha}{\alpha}. \qquad (18)$$

This shows that for large regularization parameters, the condition number is close to one and convergence is fast. On the other hand, for small regularization parameters, the condition number gets very large, even if $X$ is well-conditioned. On first glance, it seems paradoxical that a small modification of the objective function can change the convergence behavior completely. But for a small regularization constant, the objective function has a very flat optimum instead of being constant in these directions. Finding the *exact* optimum is indeed very hard. On the other hand, the optimization is dominated by the unregularized part of the objective function. Therefore, the iterates of the optimization algorithm will be close to an optimum of the unregularized objective function. Since the regularization term is only small, the iterates already correspond to good models according to the objective function.

## 5 Experimental Results

In this section, we validate the theoretical results on two classification tasks. The first one is the well-known USPS task for handwritten digit recognition. The second task, IAM, is a large-scale continuous handwriting recognition task with real-life data. Our main interest is the large-scale task, since this is a task for which log-linear models are especially useful.

Table 1: Results on the USPS task for different feature transformations and regularization parameters $\alpha$. The columns "separation" and "termination" list the number of passes through the dataset until separation of the training data, respectively the termination of the optimization algorithm.

| Preprocessing | $\alpha N$ | Train error (%) | Separation | Termination |
|---|---|---|---|---|
| Whitening and mean norm. | 0.0 | 0.0 | 21 | 66 |
| Mean and variance norm. | 0.0 | 0.0 | 61 | 116 |
| None | 0.0 | 0.0 | 356 | 513 |
| None | 0.01 | 0.03 | - | 731 |
| None | 0.1 | 0.43 | - | 358 |
| None | 1.0 | 2.08 | - | 174 |
| None | 10.0 | 4.29 | - | 100 |

## 5.1 Handwritten Digit Recognition

The USPS dataset[2] consists of 7291 training and 2007 test images from ten classes of handwritten digits. We trained a log-linear classifier directly on the whole image with $16 \times 16$ pixels.

We used the L-BFGS algorithm for optimization, which is considered as the best algorithm for log-linear training. For all experiments, we used a a backtracking line search and a history length of ten, which is a standard value given in literature [14, 21]. We stopped the optimization, when the relative change in the objective was below $\epsilon = 10^{-5}$, i.e.

$$(\mathcal{F}(\Lambda_{k-1}) - \mathcal{F}(\Lambda_k))/\mathcal{F}(\Lambda_k) < \epsilon \ . \tag{19}$$

Table 1 contains the results on the USPS task. The results reflect our analysis of the condition number. Without normalizing mean and variance, the optimization problem is not well-conditioned. It requires more than 500 passes through the dataset until the termination criterion is reached. The optimization takes even longer, when a very small non-zero regularization constant is used. This is what we expected by analyzing the condition number – the objective function has a very flat optimum, which slows down convergence. On the other hand, for higher regularization parameters, the optimization is much faster. We applied the normalizations only to the unregularized models, because the feature transformations affect the regularization term. Therefore, results with regularization are not comparable when feature transformations are applied. The mean and variance normalization reduced the computational costs greatly, from 513 to 116 iterations. The application of the whitening transformation further reduced the number of iterations to 66. Often, the classification error on the training data reaches its minimum before the optimization algorithm terminates, so one might argue that it is not necessary to run the optimization until the termination criterion is reached. The USPS training data is linearly separable and for all unregularized trainings, a zero classification error on the training set is reached. It turns out that the effect of the feature transformations is even stronger when the number of iterations until the training data is separated is compared (see Table 1).

## 5.2 Handwritten Text Recognition

Our second task is the IAM handwriting database [15]. In contrast to USPS, where single images are classified into a small number of classes, IAM defines a continuous handwriting recognition task with unrestricted vocabulary, and is therefore much harder. The corpus has a predefined subdivision into training, development, and testing folds. The training fold contains lines of handwritten text with 53k words in total. With our feature extraction, this corresponds to $3,592,006$ observations. The development and test fold contain 9k respectively 25k words. The IAM database is a *large-scale learning problem* in the sense that it is not feasible to run the optimization until convergence [2] and the test error is strongly influenced by the optimization accuracy.

### 5.2.1 Baseline Model

For our baseline model, we use the conventional generative approach of a statistical classifier based on hidden Markov models (HMMs) with Gaussian Mixture models (GMMs) as emission probabilities. The generative classifier maps an observation sequence $x_1^T = (x_1, \ldots, x_T) \in \mathcal{X}$ to a word

sequence $\hat{w}_1^N = (\hat{w}_1, \ldots, \hat{w}_N) \in \mathcal{W}$ according to Bayes rule:

$$r : \mathcal{X} \to \mathcal{W}, \quad x_1^T \mapsto \hat{w}_1^N = \underset{w_1^N \in \mathcal{W}}{\mathrm{argmax}}\, p_\theta(w_1^N)^\gamma p_\theta(x_1^T | w_1^N) \,. \tag{20}$$

The prior probability $p_\theta(w_1^N)$ is a smoothed trigram *language model* trained on the reference of the training data and the three additional text corpora Lancaster-Oslo-Bergen, Brown, and Wellington, as proposed in [1]. The language model scale $\gamma > 0$ has been optimized on the development set. The *visual model* $p_\theta(x_1^T | w_1^N)$ is defined by an HMM, which is composed of submodels for each character in the word sequence. In total there are 78 characters, which are modeled by five-state left-to-right HMMs, resulting in 390 distinct states plus one state for the whitespace model. The emission probabilities of the HMM are modeled by GMMs with a single shared covariance matrix. The parameters of the visual model are optimized according to the *maximum likelihood criterion* with the expectation-maximization (EM) algorithm and a splitting procedure. We obtained best results with 25k mixture components in total. We only used basic deslanting and size normalization for feature preprocessing, as it is commonly applied in handwriting recognition. An image slice was extracted at every position. Seven features in a sliding window were concatenated and projected to a thirty dimensional vector by a principal component analysis (PCA). The recognition lexicon consists of the 50k most frequent words in the language model training data. The generative baseline system achieves a word error rate (WER) of 32.8% on the development set and 39.4% on the test set, similar to the results of the GMM/HMM-baseline systems by [1, 6, 5].

### 5.2.2 Hybrid LL/HMM Recognition System

The main component of the visual model of our baseline system is the GMM for the emission probabilities $p_\theta(x|s)$. Analogous to the use of neural network outputs by [6], we build a hybrid LL/HMM recognition system by deriving the emission probabilities via $p_\Lambda(x|s) = p_\Lambda(s|x)p(x)/p(s)$. The prior probability $p(s)$ can be estimated easily as the relative frequency, and $p(x)$ can be discarded in recognition without changing the maximizing word sequence.

We used our baseline system for generating a state alignment, i.e. an assignment of the feature vectors to an HMM state, and then trained a log-linear model on the resulting training sample $(x_t, s_t)_{t=1,\ldots,T}$ analogous to the setup on USPS. Note that the training of the log-linear model is conceptually exactly the same as for USPS and our convergence analysis applies.

On large-scale tasks as IAM, it is not practicable to run the optimization until convergence as on USPS. Instead, we assume a limited training budget for all experiments, which allows for performing 200 iterations, and compare the resulting classifiers. This procedure corresponds to the characterization of large-scale learning tasks by Bottou and Bousquet [2].

The performance of a linear classifier on a complex task as IAM is quite limited. Therefore, we used polynomial feature spaces of degree one $(d = 30)$, two $(d = 495)$ and three $(d = 5455)$, corresponding to polynomial kernels. In contrast to USPS, where the classification error on the training data without regularization was zero, on IAM, the state-classification error on the training data ranges from forty to sixty percent. Thus, the impact of regularization on the performance of the classifier is only minor. In preliminary experiments, we obtained almost no improvements by regularization. Therefore, we report only the results without regularization.

### 5.2.3 Results

The results on the IAM database (see Table 2) are again in accordance to our theoretical analysis. The first-order features are already decorrelated, but without mean and variance normalization, the convergence is slower, resulting in a worse WER on the development and test set. The difference is moderate, when the parameters are initialized with zero, corresponding to a uniform distribution. In a next experiment we initialized all parameters randomly with plus or minus one. This results in a huge degradation for the unnormalized features and – with exactly the same random initialization – has only a minor impact when normalized features are used. The differences are even larger for the second-order experiments. This can be expected, since mean and variance take on more extreme values when the features are squared, and the features are correlated. For the zero initialization, the improvement from mean and variance normalization is only moderate in WER. For the unnormalized features and random initialization, the optimization did not lead to a usable

Table 2: Results on the IAM database for polynomial feature spaces of degree $m \in \{1, 2, 3\}$ with different initializations and preprocessings.

| $m$ | Preprocessing | Initialization | WER / dev set (%) | WER / test set (%) |
|---|---|---|---|---|
| 1 | none | zero / random | 49.9 / 68.3 | 60.1 / 75.5 |
| 1 | mean and var. norm. | zero / random | 49.7 / 48.9 | 58.9 / 58.5 |
| 2 | none | zero / random | 32.4 / >100.0 | 40.2 / >100.0 |
| 2 | mean and var. norm. | zero / random | 30.2 / 34.4 | 38.5 / 41.3 |
| 2 | mean and var. norm. | 1st order | 26.8 | 33.1 |
| 2 | whitening and mean norm. | zero / random | 25.1 / 25.9 | 31.6 / 32.3 |
| 3 | mean and var. norm. | 2nd order | 23.0 | 27.4 |

model for recognition at all. Fastest convergence and best results are obtained by the application of the whitening transformation to the features. In addition, the influence of the initialization is the smallest in this case. Because of the high dimension of the third-order features, the estimation of the whitening transformation itself is already computationally very expensive. Therefore, we only performed a mean and variance normalization of the third-order features, but initialized the models incrementally from first to second to third-order features. In this manner, we obtain our best result of 27.4% WER, which is a drastic improvement over the generative baseline system (39.4% WER).

Our hybrid LL/HMM system outperforms other systems based on HMMs with comparable preprocessing. Bertolami and Bunke [1] obtain 32.9% WER with an ensemble-based HMM approach. Dreuw et al. [5] obtain 30.0% WER with discriminatively trained GMMs and 29.0% WER with an additional discriminative adaptation method. The system of Graves [7], which has a completely different architecture based on recurrent neural networks, outperforms our system with 25.9% WER. The best published result of 21.2% WER on the IAM database is by España-Boquera et al. [6], who use several specialized neural networks for preprocessing.

## 6 Discussion

In this paper, we presented a novel convergence analysis for the optimization of the parameters of log-linear models. Our main results are first that the convergence of gradient-based optimization algorithms depends on the eigenvalues of the uncentered empirical covariance matrix. For this derivation we assumed that the quadratic term of the objective function at the optimum behaves similar as at the initialization. Second, we analyzed the eigenvalues of the covariance matrix. According to this analysis, it is important to normalize mean and variances of the features. Best convergence behavior can be expected when, in addition, the features are decorrelated.

Interestingly, the same result is obtained by LeCun et al. [11] for neural network training, but their analysis differs from ours in a number of aspects. First, LeCun et al. consider a simpler loss function. In contrast to our analysis, they assume that all components of the observations have identical mean and variance and that the components are independent. Furthermore, they fix the ratio of the number of model parameters and the number of training observations. The derivation of the spectrum of the Hessian is then performed in the limit of infinite training data, leading to a continuous spectrum. This approach is more suited for the analysis of online learning. In the case of batch learning, the training data as well as the model size is fixed.

We verified our findings on two handwriting recognition tasks and found that the theoretical analysis predicted the observed convergence behavior very well. On IAM, a real-life dataset for continuous handwriting recognition, our log-linear system outperforms other systems with comparable architecture and preprocessing. This is remarkable, because we use a generic and conceptually simple method, which is simple to implement and allows for reproducing experimental results easily.

An interesting point for future work is the use of approximate decorrelation techniques, e.g. by assuming a structure for the covariance matrix. This will be useful for very high-dimensional features for which the estimation of the whitening transformation is not feasible.

## Footnotes

[1]Recall that the condition number of a positive definite matrix $A$ is the ratio of its largest and its smallest eigenvalues: $\kappa(A) = \lambda_{\max}(A)/\lambda_{\min}(A)$

[2]ftp://ftp.kyb.tuebingen.mpg.de/pub/bs/data/

# References

[1] Bertolami, R., Bunke, H.: HMM-based Ensamble Methods for Offline Handwritten Text Line Recognition. Pattern Recogn. 41, 3452–3460 (2008)

[2] Bottou, L., Bousquet, O.: The tradeoffs of large scale learning. In: Advances in Neural Information Processing Systems. pp. 161–168 (2008)

[3] Boyd, S., Vandenberghe, L.: Convex Optimization. Cambridge University Press (2004)

[4] Darroch, J., Ratcliff, D.: Generalized Iterative Scaling for Log-Linear Models. Ann. Math. Stat. 43(5), 1470–1480 (1972)

[5] Dreuw, P., Heigold, G., Ney, H.: Confidence- and Margin-Based MMI/MPE Discriminative Training for Off-Line Handwriting Recognition. Int. J. Doc. Anal. Recogn. pp. 1–16 (2011)

[6] España-Boquera, S., Castro-Bleda, M., Gorbe-Moya, J., Zamora-Martinez, F.: Improving Offline Handwritten Text Recognition with Hybrid HMM/ANN Models. IEEE Trans. Pattern Anal. Mach. Intell. 33(4), 767 –779 (april 2011)

[7] Graves, A., Liwicki, M., Fernandez, S., Bertolami, R., Bunke, H., Schmidhuber, J.: A Novel Connectionist System for Unconstrained Handwriting Recognition. IEEE Trans. Pattern Anal. Mach. Intell. 31(5), 855–868 (May 2009)

[8] Horn, R., Johnson, C.: Topics in Matrix Analysis. Cambridge University Press (1994)

[9] Horn, R., Johnson, C.: Matrix Analysis. Cambridge University Press (2005)

[10] Lafferty, J., McCallum, A., Pereira, F.: Conditional random fields: Probabilistic models for segmenting and labeling sequence data. In: Proceedings of the 18th International Conference on Machine Learning. pp. 282–289 (2001)

[11] LeCun, Y., Kanter, I., Solla, S.: Second order properties of error surfaces: Learning time and generalization. In: Advances in Neural Information Processing Systems. pp. 918–924. Morgan Kaufmann Publishers Inc. (1990)

[12] Liu, D., Nocedal, J.: On the Limited Memory BFGS Method for Large-Scale Optimization. Math. Program. 45(1), 503–528 (1989)

[13] Luenberger, D., Ye, Y.: Linear and Nonlinear Programming. Springer Verlag (2008)

[14] Malouf, R.: A comparison of algorithms for maximum entropy parameter estimation. In: Proceedings of the Sixth Conference on Natural Language Learning. pp. 49–55 (2002)

[15] Marti, U., Bunke, H.: The IAM-Database: An English Sentence Database for Offline Handwriting Recognition. Int. J. Doc. Anal. Recogn. 5(1), 39–46 (2002)

[16] McCallum, A., Freitag, D., Pereira, F.: Maximum entropy markov models for information extraction and segmentation. In: Proceedings of the 17th International Conference on Machine Learning. pp. 591–598 (2000)

[17] Minka, T.: Algorithms for maximum-likelihood logistic regression. Tech. rep., Carnegie Mellon University (2001)

[18] Nocedal, J., Wright, S.: Numerical Optimization. Springer (1999)

[19] Notay, Y.: Solving positive (semi)definite linear systems by preconditioned iterative methods. In: Preconditioned Conjugate Gradient Methods, Lecture Notes in Mathematics, vol. 1457, pp. 105–125. Springer (1990)

[20] Salakhutdinov, R., Roweis, S., Ghahramani, Z.: On the convergence of bound optimization algorithms. In: Uncertainty in Artificial Intelligence. vol. 19, pp. 509–516 (2003)

[21] Sha, F., Pereira, F.: Shallow parsing with conditional random fields. In: Proceedings of the 2003 Conference of the North American Chapter of the Association for Computational Linguistics on Human Language Technology. pp. 134–141 (2003)

[22] Sutton, C., McCallum, A.: An introduction to conditional random fields for relational learning. In: Getoor, L., Taskar, B. (eds.) Introduction to Statistical Relational Learning. MIT Press (2007)

